# Connectionism for Music and Audition

**Andreas S. Weigend**
Department of Computer Science
and Institute of Cognitive Science
University of Colorado
Boulder, CO 80309-0430

## Abstract

This workshop explored machine learning approaches to 3 topics: (1) finding structure in music (analysis, continuation, and completion of an unfinished piece), (2) modeling perception of time (extraction of musical meter, explanation of human data on timing), and (3) interpolation in timbre space.

In recent years, NIPS has heard neural networks generate tunes and harmonize chorales. With a large amount of music becoming available in computer readable form, real data can be used to train connectionist models. At the beginning of this workshop, **Andreas Weigend** focused on architectures to capture structure on multiple time scales. J. S. Bach's last (unfinished) fugue from *Die Kunst der Fuge* served as an example (Dirst & Weigend, 1994).[1] The prediction approach to continuation and completion, as well as to modeling expectations, can be characterized by the question "What's next?". Moving to time as the primary medium of musical communication, the inquiry in music perception and cognition shifted to the question "When next?".

In other words, so far we have considered patterns *in* time. They assume prior identification and subsequent processing of events. **Bob Port**, coming from the speech community, considered patterns *of* time, discussing timing in linguistic polyrhythms (e.g., *hot cup of tea*). He also drew parallels between timing in Japanese language and timing in music, supporting the hypothesis that perceptional rhythms entrain attentional rhythms. As a mechanism for entrainment, **Devin McAuley** presented adaptive oscillators: the oscillators adapt their frequencies such that their "firing" coincides with the beat of the music (McAuley, 1994).

As the beat can be viewed as entrainment of an individual oscillator, the meter can be viewed as entrainment of multiple oscillators. **Ed Large** described human perception of metrical structure in analogy to two pendulum clocks that synchronize their motions by hanging on the same wall. An advantage of these entrainment

approaches (which focus on time as time) over traditional approaches (which focus on music notation and treat time symbolically) is their ability to model phenomena in music performance, such as expressive timing.

Taking a Gibsonian perspective, **Fred Cummins** emphasized the relevance of eco­logical constraints on audition: perceptually relevant features are not easily spotted in the wave form or the spectrum. Among the questions he posed were: what "higher-order" features might be useful for audition, and whether recurrent net­works could be useful to extract such features.

The last contribution also addressed the issue of representation, but with sound synthesis in mind: wouldn't a musician like to control sound in a perceptually rele­vant space, rather than fiddling with non-intuitive coefficients of an FM-algorithm? Such a space was constructed with human input: subjects were asked to similarity-judge sounds from different instruments (normalized in pitch, duration and volume). Multidimensional scaling was used to define a low-dimensional sub-space keeping the distance relations. **Michael Lee** first trained a network to find a map from timbre space to the space of the first 33 harmonics (Lee, 1994). He then used the network to generate rich new sounds by interpolating in this perceptually relevant space, through physical gestures, such as from a data glove, or through an interface musicians might be comfortable with, such as a cello.

The discussion turned to the importance of working with perceptually adequate, "ecologically sound" representations (e.g., by using a cochlea model as pre-processor, or a speech model as post-processor for sonification applications). Finally, to probe human cognition, we discussed synthetic sounds, designed to reveal fundamental characteristics of the auditory system, independent of our daily experience. Return­ing to the title, the workshop turned out to be problem driven: people presented a problem or a finding and searched for a solution—connectionist or otherwise—rather than applying canned connectionist ideas to music and cognition.

I thank the speakers, Fred Cummins (fcummins@indiana.edu), Ed Large (large@cis.ohio-state.edu), Michael Lee (lee@cnmat.berkeley.edu), Devin McAuley (mcauley@cs.indiana.edu), Robert Port (port@indiana.edu), as well as all partici­pants. I also thank Tom Ngo (ngo@interval.com) for sending me the notes he took at the workshop, and Eckhard Kahle (kahle@ircam.fr) for discussing this summary.

## Footnotes

[1] This fugue is available via anonymous ftp from `ftp.santafe.edu` as data set `F.dat` of the Santa Fe Time Series Analysis and Prediction Competition.

## References

Dirst, M., and A. S. Weigend (1994) "Baroque Forecasting: On Completing J. S. Bach's Last Fugue." In *Time Series Prediction: Forecasting the Future and Understanding the Past*, edited by A. S. Weigend and N. A. Gershenfeld, pp. 151–172. Addison-Wesley.

Lee, M., and D. Wessel (1992) "Connectionist Models for Real-Time Control of Syn­thesis and Compositional Algorithms." In *Proceedings of the International Com­puter Music Conference*, pp. 277–280. San Francisco, CA: International Computer Music Association.

McAuley, J. D. (1994) "Finding metrical structure in time." In *Proceedings of the 1993 Connectionist Models Summer School*, edited by M. C. Mozer, P. Smolensky, D. S. Touretzky, J. L. Elman and A. S. Weigend, pp. 219–227. Lawrence Erlbaum.